# Generalised Propagation for Fast Fourier Transforms with Partial or Missing Data

**Amos J Storkey**
School of Informatics, University of Edinburgh
5 Forrest Hill, Edinburgh UK
*a.storkey@ed.ac.uk*

## Abstract

Discrete Fourier transforms and other related Fourier methods have been practically implementable due to the fast Fourier transform (FFT). However there are many situations where doing fast Fourier transforms without complete data would be desirable. In this paper it is recognised that formulating the FFT algorithm as a belief network allows suitable priors to be set for the Fourier coefficients. Furthermore efficient generalised belief propagation methods between clusters of four nodes enable the Fourier coefficients to be inferred and the missing data to be estimated in near to $\mathcal{O}(n \log n)$ time, where $n$ is the total of the given and missing data points. This method is compared with a number of common approaches such as setting missing data to zero or to interpolation. It is tested on generated data and for a Fourier analysis of a damaged audio signal.

## 1 Introduction

The fast Fourier transform is a fundamental component in any numerical toolbox. Commonly it is thought of as a deterministic transformation from data to Fourier space. It relies on regularly spaced data, ideally of length $2^{h_i}$ for some $h_i$ in each dimension $i$. However there are many circumstances where Fourier analysis would be useful for data which does not take this form. The following are a few examples of such situations:

- There is temporary/regular instrument failure or interruption.
- There are scratches on media, such as compact disks.
- Missing packets occur in streamed data.
- Data is not $2^k$ in length or is from e.g. irregularly shaped image patches.
- There is known significant measurement error in the data.
- Data is quantised, either in Fourier domain (e.g. jpeg) or data domain (e.g. integer storage).

Setting missing values to zeros or using interpolation will introduce various biases which will also affect the results; these approaches can not help in using Fourier information to help restore the missing data.

Prior information is needed to infer the missing data or the corresponding Fourier components. However to be practically useful inference must be fast. Ideally we want techniques which scale close to $\mathcal{O}(n \log n)$.

The FFT algorithm can be described as a belief network with deterministic connections where each intermediate node has two parents and two children (a form commonly called the butterfly net). The graphical structure of the FFT has been detailed before in a number of places. See [1, 5] for examples. Prior distributions for the Fourier coefficients can be specified. By choosing a suitable cluster set for the network nodes and doing generalised propagation using these clusters, reasonable inference can be achieved. In the case that all the data is available this approach is computationally equivalent to doing the exact FFT.

There have been other uses of belief networks and Bayesian methods to improve standard transforms. In [2], a hierarchical prior model of wavelet coefficients was used with some success. Other authors have recognised the problem of missing data in hierarchical systems. In [6] the authors specify a multiscale stochastic model, which enables a scale recursive description of a random process. Inference in their model is propagation within a tree structured belief network. FFT related Toeplitz methods combined with partition inverse equations are applicable for inference in grid based Gaussian process systems with missing data [9].

## 2    Fast Fourier Transform

### 2.1    The FFT Network

From this point forward the focus will be on the one dimensional fast Fourier transform. The FFT utilises a simple recursive relationship in order to implement the discrete Fourier transform in $\mathcal{O}(n \log n)$ time for $n = 2^h$ data points. For $W = \exp(-2\pi i/n)$, the $k$th Fourier coefficient $F_k$ is given by

$$F_k \overset{\text{def}}{=} \sum_{j=0}^{n-1} W^{kj} x_j = \sum_{j=0}^{n/2-1} W^{2kj} x_{2j} + \sum_{j=0}^{n/2-1} W^{(2j+1)k} x_{2j+1} = F_k^e + W^k F_k^o \qquad (1)$$

where $F_k^e$ denotes the $k$th component of the length $n/2$ Fourier transform of the even components of $x_j$. Likewise $F_k^o$ is the same for the odd components. The two new shorter Fourier transforms can be split in the same way, recursively down to the transforms of length 1 which are just the data points themselves. It is also worth noting that $F_k^e$ and $F_k^o$ are in fact used twice, as $F_{k+n/2} = F_k^e - W^k F_k^o$. The inverse FFT uses exactly the same algorithm as the FFT, but with conjugate coefficients.

This recursive algorithm can be drawn as a network of dependencies, using the inverse FFT as a generative model; it takes a set of Fourier components and creates some data. The usual approach to the FFT is to shuffle the data into reverse bit order ($x_i$ for binary $i = 010111$ is put in position $i' = 111010$; see [8] for more details). This places data which will be combined in adjacent positions. Doing this, we get the belief network of Figure 1a as a representation of the dependencies. The top row of this figure gives the Fourier components in order, and the bottom row gives the bit reversed data. The intermediate nodes are the even and odd Fourier coefficients at different levels of recursion.

### 2.2    A Prior on Fourier Coefficients

The network of Figure 1a, combined with (1), specifies the (deterministic) conditional distributions for all the nodes below the top layer. However no prior distribution is currently set for the top nodes, which denote the Fourier coefficients. In general little prior phase information is known, but often there might be some

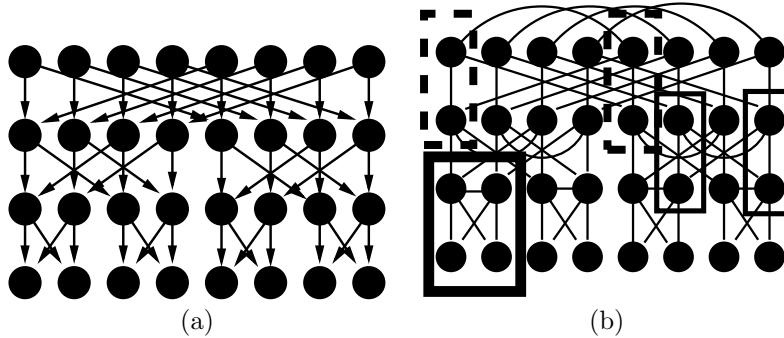

<div style="text-align:center">(a)            (b)</div>

Figure 1: (a) The belief network corresponding to the fast Fourier transform. The top layer are the Fourier components in frequency order. The bottom layer is the data in bit reversed order. The intermediate layers denote the partial odd and even transforms that the algorithm uses. (b) The moralised undirected network with three clusters indicated by the boxes. All nodes surrounded by the same box type form part of the same cluster.

expected power spectra. For example we might expect a $1/f$ power spectra, or in some circumstances empirical priors may be appropriate. For simplicity we choose independent complex Gaussian[1] priors on each of the top nodes. Then the variance of each prior will represent the magnitude of the expected power of that particular coefficient.

## 3 Inference in the FFT Network

Suppose that some of the data which would be needed to perform the FFT is missing. Then we would want to infer the Fourier coefficients based on the data that was available. The belief network of Figure 1a is not singly connected and so exact propagation methods are not appropriate. Forming the full Gaussian distribution of the whole network and calculating using that is too expensive except in the smallest situations. Using exact optimisation (eg conjugate gradient) in the conditional Gaussian system is $\mathcal{O}(n^2)$, although a smaller number of iterations of conjugate gradient can provide a good approximation. Marrying parents and triangulating the system will result in a number of large cliques and so junction tree methods will not work in a reasonable time.

### 3.1 Loopy Propagation

Loopy propagation [7, 10, 3] in the FFT network suffers from some serious deficits. Experiments with loopy propagation suggest that often there are convergence problems in the network, especially for systems of any significant size. Sometimes adding additional jitter and using damping approaches (see e.g. [4]) can help the system to converge, but convergence is then very slow. Intuitively the approximation given by loopy propagation fails to capture the moralisation of the parents, which, given the deterministic network, provides strong couplings. Note that when the system does converge the mean inferred values are correct [11], but the variances are significantly underestimated.

# 4 Generalised Belief Propagation for the FFT

In [14] the authors show that stationary points of loopy propagation between nodes of a Markov network are minimisers of the Bethe free energy of the probabilistic system. They also show that more general propagation procedures, such as propagation of information between clusters of a network correspond to the minimisation of a more general Kikuchi free energy of which The Bethe free energy is a special case.

To overcome the shortfalls of belief propagation methods, a generalised belief propagation scheme is used here. The basic problem is that there is strong dependence between parents of a given node, and the fact that the values for those two nodes are fully determined by the two children but undetermined by only one. Hence it would seem sensible to combine these four nodes, the two parents and two children, together into one cluster. This can be done for all nodes at all levels, and we find that the cluster separator between any two clusters consists of at most one node. At each stage of propagation between clusters only the messages (in each direction) at single nodes need to be maintained.

The procedure can be summarised as follows: Start with the belief network of Figure 1a and convert it to an undirected network by moralisation (Figure 1b). Then we identify the clusters of the graph, which each consist of four nodes as illustrated by the boxes in Figure 1b. Each cluster consists of two common parents and their common children. Each node not in the extreme layers is also a separator between two clusters. Building a network of clusters involves creating an edge for each separator. From Figure 1 it can be seen that this network will have undirected loops. Hence belief propagation in this system will not be exact. However it will be possible to iteratively propagate messages in this system. Hopefully the iteration will result in an equilibrium being reached which we can use as an approximate inference for the marginals of the network nodes, although such convergence is not guaranteed.

## 4.1 Propagation Equations

This section provides the propagation messages for the approach described above. For simplicity, and to maintain symmetry we use an update scheme where messages are first passed down from what were the root nodes (before moralisation) to the leaf nodes, and then messages are passed up from the leaf to the root nodes. This process is then iterated. The first pass down the network is data independent and can be precomputed.

### 4.1.1 Messages Propagating Down

The Markov network derived from a belief network has the potentials of each cluster defined by the conditional probability of all the child nodes in that cluster given their parents. Two adjoining clusters of the network are illustrated in Figure 2a. All the cluster interactions in the network have this form, and so the message passing described below applies to all the nodes.

The message $\rho_4 \equiv N(\mu_4^+, \sigma_4^+)$ is defined to be that passed down from some cluster $C_1$ containing nodes $y_1$, $y_2$ (originally the parents) and $y_3$, $y_4$ (originally the children) to the cluster below: $C_2 = (y_4, y_5, y_6, y_7)$, with $y_6$ and $y_7$ the children. $\mu_4^+$ is the message mean, and $\sigma_4^+$ is the covariance. The message is given by the marginal of the cluster potential multiplied by the incoming messages from the other nodes. The standard message passing scheme can be followed to get the usual form of results for Gaussian networks [7, 11].

Suppose $\lambda_3(y_3) = N(y_3; \mu_3^-, \sigma_3^-)$ is the message passing up the network at node 3, whereas $\rho_1(y_1) = N(y_1; \mu_1^+, \sigma_1^+)$ and $\rho_2(y_2) = N(y_2; \mu_2^+, \sigma_2^+)$ are the messages passing down the network at nodes 1 and 2 respectively. Here we use the notation of [7] and use $\sigma$ to represent variances. Defining[2]

$$\Sigma_A = B_1 \begin{pmatrix} \sigma_1^+ & 0 \\ 0 & \sigma_2^+ \end{pmatrix} B_1^\dagger, \quad \boldsymbol{\mu}_A = B_1 \begin{pmatrix} \mu_1^+ \\ \mu_2^+ \end{pmatrix} \quad \text{where} \quad B_1 = \begin{pmatrix} b_{31} & b_{32} \\ b_{41} & b_{42} \end{pmatrix} \quad (2)$$

are the connection coefficients derived from (1), and

$$\Sigma_D^{-1} = \Sigma_A^{-1} + \begin{pmatrix} 1/\sigma_3^- & 0 \\ 0 & 0 \end{pmatrix}, \quad \boldsymbol{\mu}_D = \Sigma_D \left( \Sigma_A^{-1} \boldsymbol{\mu}_A + \begin{pmatrix} \mu_3^- / \sigma_3^- \\ 0 \end{pmatrix} \right) \quad (3)$$

allows us to write the downward message as

$$\mu_4^+ = (\boldsymbol{\mu}_D)_2 \text{ and } \Sigma_4^+ = (\Sigma_D)_{22}. \quad (4)$$

### 4.1.2   Messages Propagating Up

In the same way we can calculate the messages which are propagated up the network. The message $\lambda_4 = N(\mu_4^-, \sigma_4^-)$ passed up from cluster $C_2$ to cluster $C_1$ is given by

$$\mu_4^- = (\boldsymbol{\mu}_U)_1 \text{ and } \Sigma_4^- = (\Sigma_U)_{11} \text{ where } B_2 = \begin{pmatrix} b_{64} & b_{65} \\ b_{74} & b_{75} \end{pmatrix}, \text{ and} \quad (5)$$

$$\Sigma_B = (B_2^{-1}) \text{diag}(\sigma_6^-, \sigma_7^-)(B_2^{-1})^\dagger, \quad \boldsymbol{\mu}_B = B_2^{-1}(\mu_6, \mu_7)^T, \quad (6)$$

$$\Sigma_U^{-1} = \Sigma_B^{-1} + \text{diag}(0, 1/\sigma_5^+), \quad \boldsymbol{\mu}_U = \Sigma_U(\Sigma_B^{-1} \boldsymbol{\mu}_B + \text{diag}(0, 1/\sigma_5^+)(0, \mu_5^+)^T) \quad (7)$$

All the other messages follow by symmetry.

### 4.1.3   Calculation of the Final Marginals

The approximate posterior marginal distributions are given by the product of the $\lambda$ and $\rho$ messages. Hence the posterior marginal at each node $k$ is also a Gaussian distribution with variance and mean given by

$$\sigma_k = \left( \frac{1}{\sigma_k^-} + \frac{1}{\sigma_k^+} \right)^{-1} \text{ and } \mu_k = \sigma_k \left( \frac{\mu_k^-}{\sigma_k^-} + \frac{\mu_k^+}{\sigma_k^+} \right) \text{ respectively.} \quad (8)$$

## 4.2   Initialisation

The network is initialised by setting the $\lambda$ messages at the leaf nodes to be $N(x, 0)$ for a node known to take value $x$ and $N(0, \infty)$ for the missing data. All the other $\lambda$ messages are initialised to $N(0, \infty)$. The $\rho$ message at a given root node is set to the prior at that root node. No other $\rho$ messages need to be initialised as they are not needed before they are computed during the first pass. Computationally, we usually have to add a small jitter term network noise, and represent the infinite variances by large numbers to avoid numeric problems.

## 5   Demonstrations and Results

In all the tests in this section the generalised propagation converged in a small number of iterations without the need to resort to damping. First we analyse the simple case where the variances of the Fourier component priors have a $1/k$ form where $k$ is the component number (i.e., frequency). To test this scenario, a set of

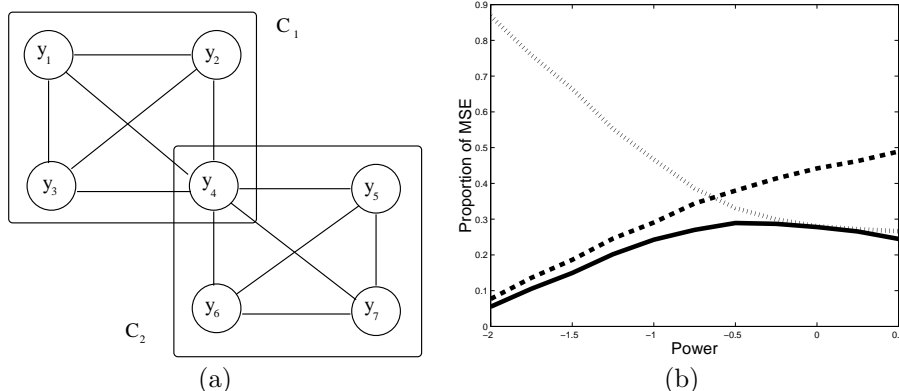

(a)                                                                  (b)

Figure 2: (a) Two clusters $C_1$ and $C_2$. All the clusters in the network contain four nodes. Each node is also common to one other cluster. Hence the interaction between any two connected clusters is of the form illustrated in this figure. (b) How the weighted mean square error varies for spectra with different power laws ($f^{\text{Power}}$). The filled line is the belief network approach, the dashed line is linear interpolation, the dotted line uses mean-valued data.

|      | Mean fill | Linear | Spline | BN    |
|------|-----------|--------|--------|-------|
| MSE  | 0.072     | 0.045  | 9.9    | 0.037 |
| WMSE | 1.6       | 0.98   | 37.7   | 0.92  |

Table 1: Comparison of methods for estimating the FFT of a $1/f$ function. 'Zero fill' replaces missing values by zero and then does an FFT. 'Linear' interpolates linearly for the missing values. 'Spline' does the same with a cubic spline. 'BN' are the results using the mean Fourier components produced by the method described in this paper.

128 complex Fourier components are generated from the prior distribution. An inverse FFT is used to generate data from the Fourier components. A predetermined set of elements is then 'lost'[3]. The remaining data is then used with the algorithm of this paper using 10 iterations of the down-up propagation. The resulting means are compared with the components obtained by replacing the missing data with zeros or with interpolated values and taking an FFT . Mean squared errors (MSE) in the Fourier components are calculated for each approach over the 100 different runs. Weighted mean squared errors (WMSE) are also calculated, where each frequency component is divided by its prior variance before averaging. The results are presented in Table 1.

The generalised belief propagation produces better results than any of the other approaches. Similar results are achieved for a number of different spectral priors. The benefits of interpolation are seen for situations where there are only low frequency components, and the zeroing approach becomes more reasonable in white noise like situations, but across a broad spread of spectral priors, the belief network approach tends to perform better. Figure 2b illustrates of how the results vary for an average of 100 runs as the power spectrum varies from $f^{-2}$ to $f^{0.5}$. Note that the approach is particularly good at the $1/f$ power spectra point, which corresponds to the form of spectra in many real life problems.

|  | Linear | Spline | BN |
|---|---|---|---|
| $1/f^4$ | $3.41 \times 10^{-8}$ | $1.72 \times 10^{-8}$ | $8.51 \times 10^{-7}$ |
| $1/f^2$ | $3.33 \times 10^{-6}$ | $9.90 \times 10^{-6}$ | $3.53 \times 10^{-6}$ |
| $1/f$ | $9.39 \times 10^{-5}$ | $5.15 \times 10^{-4}$ | $5.52 \times 10^{-5}$ |

Table 2: Testing the MSE predictive ability of the belief network approach.

|  | Zero fill | Linear | Spline | BN |
|---|---|---|---|---|
| MSE | 3.421 | 1.612 | 0.869 | 0.317 |
| WMSE | 1.96 | 0.883 | 0.465 | 0.125 |
| MSEPRED | 0.0033 | 0.0016 | 0.00085 | 0.00031 |

Table 3: Testing the ability of the belief network approach on real life audio data. The BN approach performs better than all others for both prediction of the correct spectrum and prediction of the missing data. MSE: mean squared error, WMSE: weighted mean squared error, MSEPRED: Mean squared error of the data predictor.

Next we compare approaches for filling in missing data. This time 50 runs are made on $1/f^4$, $1/f^2$ and $1/f$ power spectra. Note that ignoring periodic boundary constraints, a $1/f^2$ power spectra produces a Brownian curve for which the linear predictor is the optimal mean predictor. In this case the mean square error for the belief network propagation approach (Table 2) is close to the linear error. On smooth curves such as that produced by the $1/f^4$ noise the predictive ability of the approach (for small numbers of iterations) does not match interpolation methods. The local smoothness information is not easily used in the belief network propagation, because neighbouring points in data space are only connected at the highest level in the belief network. The approximations of loopy propagation methods do not preserve enough information when propagated over these distances. However for data such as that produced by the common $1/f$ power spectra, interpolation methods are less effective, and the belief network propagation performs well. In this situation the belief network approach outperforms interpolation. Calculations using zero values or mean estimates also prove significantly worse.

Last, tests are made on some real world audio data. A 1024 point complex audio signal is built up from a two channel sample from a short stretch of laughter. Fourier power spectra of the mean of 15 other different sections of laughter are used to estimate the prior power spectral characteristics. Randomly selected parts of the data are removed corresponding to one tenth of the whole. A belief network FFT is then calculated in the usual way, and compared with the true FFT calculated on the whole data. The results are given in Table 3. The belief network approach performs better than all other methods including linear and cubic spline interpolation.

## 6 Discussion

This paper provides a clear practical example of a situation where generalised propagation overcomes deficits in simpler propagation methods. It demonstrates how a belief network representation of the fast Fourier transform allows Fourier approaches to be used in situations where data is missing.

Kikuchi inference in the FFT belief network proves superior to many naive approaches for dealing with missing data in the calculation of Fourier transforms. It also provides methods for inferring missing data. It does this while maintaining $\mathcal{O}(n \log_2 n)$ nature of the FFT algorithm, if we assume that the number of iterations needed for convergence does not increase with data size. In practice, additional investigations have shown that this is not the case, but that the increase in the number of iterations does not scale badly. Further investigation is needed

to show exactly what the scaling is, and further documentation of the benefits of generalised propagation over loopy propagation and conjugate gradient methods is needed beyond the space available here. It might be possible that variational approximation using clusters [12] could provide another approach to inference in this system. This paper has also not considered the possibility of dependent or sparse priors over Fourier coefficients, or priors over phase information, all of which would be interesting. Formalising the extension to 2 dimensions would be straightforward but valuable, as it is likely the convergence properties would be different.

In conclusion the tests done indicate that this is a valuable approach for dealing with missing data in Fourier analysis. It is particularly suited to the types of spectra seen in real world situations. In fact loopy propagation methods in FFT networks are also valuable in many scenarios. Very recent work of Yedidia [13], shows that discrete generalised belief propagation in FFT constructions may enable the benefits of sparse decoders to be used for Reed-Solomon codes.

### Acknowledgements

This work was funded by a research fellowship from Microsoft Research, Cambridge. The author specifically thanks Erik Sudderth, Jonathan Yedidia, and the anonymous reviewers for their comments.

## Footnotes

[1]A complex Gaussian is of the form $\exp(-0.5\mathbf{x}^T C^{-1}\mathbf{x})/Z$ where $\mathbf{x}$ is complex, and $C$ is positive (semi)definite. It is a more restrictive distribution than a general Gaussian in the complex plane.

[2]The †operator is used to denote the complex conjugate transpose (adjoint).

[3]Data in positions 3 4 5 6 8 11 13 15 18 21 22 24 25 27 28 29 30 32 33 34 35 36 42 47 51 55 58 61 65 67 71 73 75 77 78 79 81 84 86 94 97 101 102 103 104 114 115 116 117 118 119 120 121 122 123 124 125 126 127 are removed. This provides a mix of loss in whole regions, but also at isolated points.

## References

[1] S.M. Aji and R.J. McEliece. The generalised distributive law. *IEEE Trans. Info. Theory*, 47(2):498–519, February 2000.

[2] C. A. Bouman and M. Shapiro. A multiscale random field model for Bayesian image segmentation. *IEEE Transactions on Image Processing*, 3(2):162–177, 1994.

[3] B. J. Frey. Turbo factor analysis. Technical Report TR-99-1, University of Waterloo, Computer Science, April 1999.

[4] T. Heskes. Stable fixed points of loopy propagation are minima of the Bethe Free Energy. In *NIPS15*, pages 343–350, 2003.

[5] F. R. Kschischang, B. J. Frey, and H. A. Loeliger. Factor graphs and the sum–product algorithm. *IEEE Trans. Info. Theory*, 47(2):498–519, February 2001.

[6] M. R. Luettgen and A. S. Willsky. Likelihood calculation for a class of multiscale stochastic models, with application to texture discrimination. *IEEE Transactions on Image Processing*, 4(2):194–207, 1995.

[7] J. Pearl. *Probabilistic Reasoning in Intelligent Systems: Networks of Plausible Inference.* Morgan Kaufmann, 1988.

[8] W. H. Press, S. A. Teukolsky, W. T. Vetterling, and B. P. Flannery. *Numerical Recipies in C*. Cambride University Press, 1988.

[9] A. J. Storkey. Truncated covariance matrices and toeplitz methods in Gaussian processes. In *ICANN99*, pages 55–60, 1999.

[10] Y. Weiss. Correctness of local probability propagation in graphical models with loops. *Neural Computation*, 12:1–41, 2000.

[11] Y. Weiss and W. T. Freeman. Correctness of belief propagation in Gaussian models of arbitrary topology. Technical Report TR UCB//CSD-99-1046, University of California at Berkeley Computer Science Department, June 1999.

[12] W. Wiegerinck and D. Barber. Variational belief networks for approximate inference. In La Poutre and Van den Henk, editors, *Proceedings of the 10th Netherlands/Belgium Conference on AI*, pages 177–183. CWI, 1998.

[13] J. S. Yedidia. Sparse factor graph representations of Reed-Solomon and related codes. Technical Report TR2003-135, MERL, January 1994.

[14] J. S. Yedidia, W. T. Freeman, and Y. Weiss. Generalised belief propagation. In *NIPS13*, pages 689–695, 2001.
